# Self-Paced Learning for Latent Variable Models

**M. Pawan Kumar**     **Benjamin Packer**     **Daphne Koller**
Computer Science Department
Stanford University
{pawan,bpacker,koller}@cs.stanford.edu

## Abstract

Latent variable models are a powerful tool for addressing several tasks in machine learning. However, the algorithms for learning the parameters of latent variable models are prone to getting stuck in a bad local optimum. To alleviate this problem, we build on the intuition that, rather than considering all samples simultaneously, the algorithm should be presented with the training data in a *meaningful* order that facilitates learning. The order of the samples is determined by how *easy* they are. The main challenge is that often we are not provided with a readily computable measure of the easiness of samples. We address this issue by proposing a novel, iterative *self-paced learning* algorithm where each iteration simultaneously selects easy samples and learns a new parameter vector. The number of samples selected is governed by a weight that is annealed until the entire training data has been considered. We empirically demonstrate that the self-paced learning algorithm outperforms the state of the art method for learning a latent structural SVM on four applications: object localization, noun phrase coreference, motif finding and handwritten digit recognition.

## 1   Introduction

Latent variable models provide an elegant formulation for several applications of machine learning. For example, in computer vision, we may have many 'car' images from which we wish to learn a 'car' model. However, the exact location of the cars may be unknown and can be modeled as latent variables. In medical diagnosis, learning to diagnose a disease based on symptoms can be improved by treating unknown or unobserved diseases as latent variables (to deal with confounding factors). Learning the parameters of a latent variable model often requires solving a non-convex optimization problem. Some common approaches for obtaining an approximate solution include the well-known EM [8] and CCCP algorithms [9, 23, 24]. However, these approaches are prone to getting stuck in a bad local minimum with high training and generalization error.

Machine learning literature is filled with scenarios in which one is required to solve a non-convex optimization task, for example learning perceptrons or deep belief nets. A common approach for avoiding a bad local minimum in these cases is to use multiple runs with random initializations and pick the best solution amongst them (as determined, for example, by testing on a validation set). However, this approach is adhoc and computationally expensive as one may be required to use several runs to obtain an accurate solution. Bengio *et al.* [3] recently proposed an alternative method for training with non-convex objectives, called curriculum learning. The idea is inspired by the way children are taught: start with easier concepts (for example, recognizing objects in simple scenes where an object is clearly visible) and build up to more complex ones (for example, cluttered images with occlusions). Curriculum learning suggests using the easy samples first and gradually introducing the learning algorithm to more complex ones. The main challenge in using the curriculum learning strategy is that it requires the identification of easy and hard samples in a given training dataset. However, in many real-world applications, such a ranking of training samples may be onerous or conceptually difficult for a human to provide — even if this additional human supervision can be provided, what is intuitively "easy" for a human may not match what is easy for the algorithm in the feature and hypothesis space employed for the given application.

To alleviate these deficiencies, we introduce *self-paced learning*. In the context of human education, self-paced learning refers to a system where the curriculum is determined by the pupil's abilities rather than being fixed by a teacher. We build on this intuition for learning latent variable models by

designing an iterative approach that simultaneously selects easy samples and updates the parameters at each iteration. The number of samples selected at each iteration is determined by a weight that is gradually annealed such that later iterations introduce more samples. The algorithm converges when all samples have been considered and the objective function cannot be improved further. Note that, in self-paced learning, the characterization of what is "easy" applies not to individual samples, but to sets of samples; *a set of samples is easy if it admits a good fit in the model space*.

We empirically demonstrate that our self-paced learning approach outperforms the state of the art algorithm for learning a recently proposed latent variable model, called latent structural SVM, on four standard machine learning applications using publicly available datasets.

## 2   Related Work

Self-paced learning is related to curriculum learning in that both regimes suggest processing the samples in a meaningful order. Bengio *et al.* [3] noted that curriculum learning can be seen as a type of continuation method [1]. However, in their work, they circumvented the challenge of obtaining such an ordering by using datasets where there is a clear distinction between easy and hard samples (for example, classifying equilateral triangles vs. squares is easier than classifying general triangles vs. general quadrilaterals). Such datasets are rarely available in real world applications, so it is not surprising that the experiments in [3] were mostly restricted to small toy examples.

Our approach also has a similar flavor to active learning, which chooses a sample to learn from at each iteration. Active learning approaches differ in their sample selection criteria. For example, Tong and Koller [21] suggest choosing a sample that is close to the margin (a "hard" sample), corresponding to *anti-curriculum learning*. Cohn *et al.* [6] advocate the use of the most uncertain sample with respect to the current classifier. However, unlike our setting, in active learning the labels of all the samples are not known when the samples are chosen.

Another related learning regime is co-training, which works by alternately training classifiers such that the most confidently labeled samples from one classifier are used to train the other [5, 17]. Our approach differs from co-training in that in our setting the latent variables are simply used to assist in predicting the target labels, which are always observed, whereas co-training deals with a semi-supervised setting in which some labels are missing.

## 3   Preliminaries

We will denote the training data as $\mathcal{D} = \{(\mathbf{x}_i, \mathbf{y}_i), \cdots, (\mathbf{x}_n, \mathbf{y}_n)\}$, where $\mathbf{x}_i \in \mathcal{X}$ are the observed variables (which we refer to as input) for the $i^{th}$ sample and $\mathbf{y}_i \in \mathcal{Y}$ are the unobserved variables (which we refer to as output), whose values are known during training. In addition, latent variable models also contain latent, or hidden, variables that we denote by $\mathbf{h}_i \in \mathcal{H}$. For example, when learning a 'car' model using image-level labels, $\mathbf{x}$ represents an image, the binary output $\mathbf{y}$ indicates the presence or absence of a car in the image, and $\mathbf{h}$ represents the car's bounding box (if present).

Given the training data, the parameters $\mathbf{w}$ of a latent variable model are learned by optimizing an objective function, for example by maximizing the likelihood of $\mathcal{D}$ or minimizing the risk over $\mathcal{D}$. Typically, the learning algorithm proceeds iteratively, with each iteration consisting of two stages: (i) the hidden variables are either imputed or marginalized to obtain an estimate of the objective function that only depends on $\mathbf{w}$; and (ii) the estimate of the objective function is optimized to obtain a new set of parameters. We briefly describe two such well-known algorithms below.

**EM Algorithm for Likelihood Maximization.**    An intuitive objective is to maximize likelihood:

$$\max_{\mathbf{w}} \sum_i \log \Pr(\mathbf{x}_i, \mathbf{y}_i; \mathbf{w}) = \max_{\mathbf{w}} \left( \sum_i \log \Pr(\mathbf{x}_i, \mathbf{y}_i, \mathbf{h}_i; \mathbf{w}) - \sum_i \log \Pr(\mathbf{h}_i | \mathbf{x}_i, \mathbf{y}_i; \mathbf{w}) \right). \quad (1)$$

A common approach for this task is to use the EM method [8] or one of its many variants [12]. Outlined in Algorithm 1, EM iterates between finding the expected value of the latent variables $\mathbf{h}$ and maximizing objective (1) subject to this expectation. We refer the reader to [8] for more details.

**CCCP Algorithm for Risk Minimization.**    Given the true output $\mathbf{y}$, we denote the user-specified risk of predicting $\hat{\mathbf{y}}(\mathbf{w})$ as $\Delta(\mathbf{y}, \hat{\mathbf{y}}(\mathbf{w}))$. The risk is usually highly non-convex in $\mathbf{w}$, and therefore very difficult to minimize. An efficient way to overcome this difficulty is to use the recently proposed latent structural support vector machine (hereby referred to as latent SSVM) formulation [9, 23] that minimizes a regularized upper bound on the risk. Latent SSVM provides a linear prediction rule of

---

**Algorithm 1** *The* EM *algorithm for parameter estimation by likelihood maximization.*

---

**input** $\mathcal{D} = \{(\mathbf{x}_1, \mathbf{y}_1), \cdots, (\mathbf{x}_n, \mathbf{y}_n)\}, \mathbf{w}_0, \epsilon$.

1:   $t \leftarrow 0$
2:   **repeat**
3:      Obtain the expectation of objective (1) under the distribution $\Pr(\mathbf{h}_i|\mathbf{x}_i, \mathbf{y}_i; \mathbf{w}_t)$.
4:      Update $\mathbf{w}_{t+1}$ by maximizing the expectation of objective (1). Specifically,
       $\mathbf{w}_{t+1} = \operatorname{argmax}_{\mathbf{w}} \sum_i \Pr(\mathbf{h}_i|\mathbf{x}_i, \mathbf{y}_i; \mathbf{w}_t) \log \Pr(\mathbf{x}_i, \mathbf{y}_i, \mathbf{h}_i; \mathbf{w})$.
5:      $t \leftarrow t + 1$.
6:   **until** Objective function cannot be increased above tolerance $\epsilon$.

---

the form $f_{\mathbf{w}}(\mathbf{x}) = \operatorname{argmax}_{\mathbf{y}\in\mathcal{Y}, \mathbf{h}\in\mathcal{H}} \mathbf{w}^\top \Phi(\mathbf{x}, \mathbf{y}, \mathbf{h})$. Here, $\Phi(\mathbf{x}, \mathbf{y}, \mathbf{h})$ is the joint feature vector. For instance, in our 'car' model learning example, the joint feature vector can be modeled as the HOG [7] descriptor extracted using pixels in the bounding box $\mathbf{h}$.

The parameters $\mathbf{w}$ are learned by solving the following optimization problem:

$$\min_{\mathbf{w}, \xi_i \geq 0} \frac{1}{2}||\mathbf{w}||^2 + \frac{C}{n}\sum_{i=1}^{n}\xi_i,$$

$$\text{s.t.} \quad \max_{h_i \in \mathcal{H}} \mathbf{w}^\top \left(\Phi(\mathbf{x}_i, \mathbf{y}_i, \mathbf{h}_i) - \Phi(\mathbf{x}_i, \hat{\mathbf{y}}_i, \hat{\mathbf{h}}_i)\right) \geq \Delta(\mathbf{y}_i, \hat{\mathbf{y}}_i) - \xi_i,$$

$$\forall \hat{\mathbf{y}}_i \in \mathcal{Y}, \forall \hat{h}_i \in \mathcal{H}, i = 1, \cdots, n. \quad (2)$$

For any given $\mathbf{w}$, the value of $\xi_i$ can be shown to be an upper bound on the risk $\Delta(\mathbf{y}_i, \hat{\mathbf{y}}_i(\mathbf{w}))$ (where $\hat{\mathbf{y}}_i(\mathbf{w})$ is the predicted output given $\mathbf{w}$). The risk function can also depend on $\hat{\mathbf{h}}_i(\mathbf{w})$; that is, it can be of the form $\Delta(\mathbf{y}_i, \hat{\mathbf{y}}_i(\mathbf{w}), \hat{\mathbf{h}}_i(\mathbf{w}))$. We refer the reader to [23] for more details.

Problem (2) can be viewed as minimizing the sum of a convex and a concave function. This observation leads to a concave-convex procedure (CCCP) [24] outlined in Algorithm 2, which has been shown to converge to a local minimum or saddle point solution [19]. The algorithm has two main steps: (i) imputing the hidden variables (step 3), which corresponds to approximating the concave function by a linear upper bound; and (ii) updating the value of the parameter using the values of the hidden variables. Note that updating the parameters requires us to solve a convex SSVM learning problem (where the output $\mathbf{y}_i$ is now concatenated with the hidden variable $\mathbf{h}_i^*$) for which several efficient algorithms exist in the literature [14, 20, 22].

---

**Algorithm 2** *The* CCCP *algorithm for parameter estimation of latent* SSVM.

---

**input** $\mathcal{D} = \{(\mathbf{x}_1, \mathbf{y}_1), \cdots, (\mathbf{x}_n, \mathbf{y}_n)\}, \mathbf{w}_0, \epsilon$.

1:   $t \leftarrow 0$
2:   **repeat**
3:      Update $\mathbf{h}_i^* = \operatorname{argmax}_{h_i \in \mathcal{H}} \mathbf{w}_t^\top \Phi(\mathbf{x}_i, \mathbf{y}_i, \mathbf{h}_i)$.
4:      Update $\mathbf{w}_{t+1}$ by fixing the hidden variables for output $\mathbf{y}_i$ to $\mathbf{h}_i^*$ and solving the corresponding SSVM problem. Specifically,
       $\mathbf{w}_{t+1} = \operatorname{argmin}_{\mathbf{w}} \frac{1}{2}||\mathbf{w}||^2 + \frac{C}{n}\sum_i \max\{0, \Delta(\mathbf{y}_i, \hat{\mathbf{y}}_i) + \mathbf{w}^\top(\Phi(\mathbf{x}_i, \hat{\mathbf{y}}_i, \hat{\mathbf{h}}_i) - \Phi(\mathbf{x}_i, \mathbf{y}_i, \mathbf{h}_i^*))\}$.
5:      $t \leftarrow t + 1$.
6:   **until** Objective function cannot be decreased below tolerance $\epsilon$.

---

## 4   Self-Paced Learning for Latent Variable Models

Our self-paced learning strategy alleviates the main difficulty of curriculum learning, namely the lack of a readily computable measure of the *easiness* of a sample. In the context of a latent variable model, for a given parameter $\mathbf{w}$, this easiness can be defined in two ways: (i) a sample is easy if we are confident about the value of a hidden variable; or (ii) a sample is easy if it is easy to predict its true output. The two definitions are somewhat related: if we are more certain about the hidden variable, we may be more certain about the prediction. They are different in that certainty does not imply correctness, and the hidden variables may not be directly relevant to what makes the output of a sample easy to predict. We therefore focus on the second definition: easy samples are ones whose correct output can be predicted easily (its likelihood is high, or it lies far from the margin).

In the above argument, we have assumed a given $\mathbf{w}$. However, in order to operationalize self-paced learning, we need a strategy for simultaneously selecting the easy samples and learning the parameter $\mathbf{w}$ at each iteration. To this end, we note that the parameter update involves optimizing an objective function that depends on $\mathbf{w}$ (for example, see step 4 of both Algorithms 1 and 2). That is,

$$\mathbf{w}_{t+1} = \underset{\mathbf{w} \in \mathbb{R}^d}{\operatorname{argmin}} \left( r(\mathbf{w}) + \sum_{i=1}^{n} f(\mathbf{x}_i, \mathbf{y}_i; \mathbf{w}) \right), \tag{3}$$

where $r(.)$ is a regularization function and $f(.)$ is the negative log-likelihood for EM or an upper bound on the risk for latent SSVM (or any other criteria for parameter learning). We now modify the above optimization problem by introducing binary variables $v_i$ that indicate whether the $i^{th}$ sample is easy or not. Only easy samples contribute to the objective function. Formally, at each iteration we solve the following mixed-integer program:

$$(\mathbf{w}_{t+1}, \mathbf{v}_{t+1}) = \underset{\mathbf{w} \in \mathbb{R}^d, \mathbf{v} \in \{0,1\}^n}{\operatorname{argmin}} \left( r(\mathbf{w}) + \sum_{i=1}^{n} v_i f(\mathbf{x}_i, \mathbf{y}_i; \mathbf{w}) - \frac{1}{K} \sum_{i=1}^{n} v_i \right). \tag{4}$$

$K$ is a weight that determines the number of samples to be considered: if $K$ is large, the problem prefers to consider only "easy" samples with a small value of $f(.)$ (high likelihood, or far from the margin). Importantly, however, the samples are tied together in the objective through the parameter $\mathbf{w}$. Therefore, no sample is considered independently easy; rather, a *set* of samples is easy if a $\mathbf{w}$ can be fit to it such that the corresponding values of $f(.)$ are small. We iteratively decrease the value of $K$ in order to estimate the parameters of a latent variable model via self-paced learning. As $K$ approaches 0, more samples are included until problem (4) reduces to problem (3). We thus begin with only a few easy examples, gradually introducing more until the entire training dataset is used.

To optimize problem (4), we note that it can be relaxed such that each variable $v_i$ is allowed to take any value in the interval $[0, 1]$. This relaxation is *tight*; that is, for any value of $\mathbf{w}$ an optimum value of $v_i$ is either 0 or 1 for all samples. If $f(\mathbf{x}_i, \mathbf{y}_i; \mathbf{w}) < 1/K$ then $v_i = 1$ yields the optimal objective function value. Similarly, if $f(\mathbf{x}_i, \mathbf{y}_i; \mathbf{w}) > 1/K$ then the objective is optimal when $v_i = 0$.

Relaxing problem (4) allows us to identify special cases where the optimum parameter update can be found efficiently. One such special case is when $r(.)$ and $f(.)$ are convex in $\mathbf{w}$, as in the latent SSVM parameter update. In this case, the relaxation of problem (4) is a biconvex optimization problem. Recall that a biconvex problem is one where the variables $\mathbf{z}$ can be divided into two sets $\mathbf{z}_1$ and $\mathbf{z}_2$ such that for a fixed value of each set, the optimal value of the other set can be obtained by solving a convex optimization problem. In our case, the two sets of variables are $\mathbf{w}$ and $\mathbf{v}$. Biconvex problems have a vast literature, with both global [11] and local [2] optimization techniques. In this work, we use alternative convex search (ACS) [2], which alternatively optimizes $\mathbf{w}$ and $\mathbf{v}$ while keeping the other set of variables fixed. We found in our experiments that ACS obtained accurate results.

Even in the general case with non-convex $r(.)$ and/or $f(.)$, we can use the alternative search strategy to efficiently obtain an approximate solution for problem (4). Given parameters $\mathbf{w}$, we can obtain the optimum $\mathbf{v}$ as $v_i = \delta(f(\mathbf{x}_i, \mathbf{y}_i; \mathbf{w}) < 1/K)$, where $\delta(.)$ is the indicator function. For a fixed $\mathbf{v}$, problem (4) has the same form as problem (3). Thus, the optimization for self-paced learning is as easy (or as difficult) as the original parameter learning algorithm.

**Self-Paced Learning for Latent SSVM.** As an illustrative example of self-paced learning, Algorithm 3 outlines the overall self-paced learning method for latent SSVM, which involves solving a modified version of problem (2). At each iteration, the weight $K$ is reduced by a factor of $\mu > 1$, introducing more and more (difficult) samples from one iteration to the next. The algorithm converges when it considers all samples but is unable to decrease the latent SSVM objective function value below the tolerance $\epsilon$. We note that self-paced learning provides the same guarantees as CCCP:
**Property:** Algorithm 3 converges to a local minimum or saddle point solution of problem (2).
This follows from the fact that the last iteration of Algorithm 3 is the original CCCP algorithm.

Our algorithm requires an initial parameter $\mathbf{w}_0$ (similar to CCCP). In our experiments, we obtained an estimate of $\mathbf{w}_0$ by initially setting $v_i = 1$ for all samples and running the original CCCP algorithm for a fixed, small number of iterations $T_0$. As our results indicate, this simple strategy was sufficient to obtain an accurate set of parameters using self-paced learning.

## 5 Experiments
We now demonstrate the efficacy of self-paced learning in the context of latent SSVM. We show that our approach outperforms the state of the art CCCP algorithm on four standard machine learning

**Algorithm 3** *The self-paced learning algorithm for parameter estimation of latent* SSVM.

---

**input** $\mathcal{D} = \{(\mathbf{x}_1, \mathbf{y}_1), \cdots, (\mathbf{x}_n, \mathbf{y}_n)\}$, $\mathbf{w}_0$, $K_0$, $\epsilon$.
1: $t \leftarrow 0$, $K \leftarrow K_0$.
2: **repeat**
3:     Update $h_i^* = \operatorname{argmax}_{h_i \in \mathcal{H}} \mathbf{w}_t^\top \Phi(\mathbf{x}_i, \mathbf{y}_i, \mathbf{h}_i)$.
4:     Update $\mathbf{w}_{t+1}$ by using ACS to minimize the objective $\frac{1}{2}\|\mathbf{w}\|^2 + \frac{C}{n}\sum_{i=1}^n v_i \xi_i - \frac{1}{K}\sum_{i=1}^n v_i$
      subject to the constraints of problem (2) as well as $\mathbf{v} \in \{0,1\}^n$.
5:     $t \leftarrow t+1$, $K \leftarrow K/\mu$.
6: **until** $v_i = 1, \forall i$ and the objective function cannot be decreased below tolerance $\epsilon$.

---

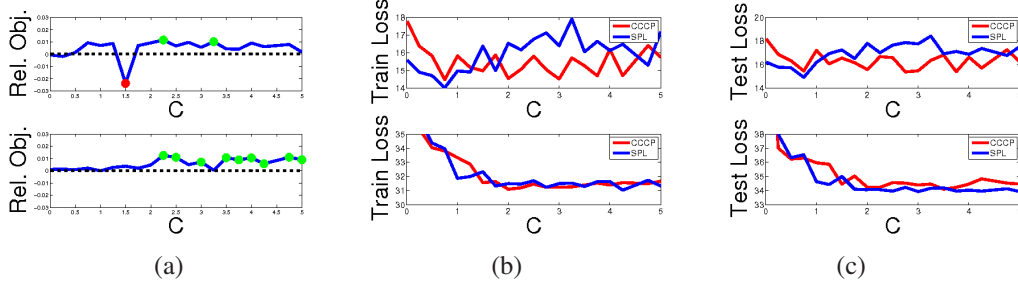

(a)                 (b)                 (c)

Figure 1: *Results for the noun phrase coreference experiment. Top:* MITRE *score. Bottom: Pairwise score. (a) The relative objective value computed as* $(obj_{cccp} - obj_{spl})/obj_{cccp}$, *where* $obj_{cccp}$ *and* $obj_{spl}$ *are the objective values of* CCCP *and self-paced learning respectively. A green circle indicates a significant improvement (greater than tolerance* $C\epsilon$*), while a red circle indicates a significant decline. The black dashed line demarcates equal objective values. (b) Loss over the training data. Minimum* MITRE *loss:* 14.48 *and* 14.02 *for* CCCP *and self-paced learning respectively; Minimum pairwise loss:* 31.10 *and* 31.03*. (c) Loss over the test data. Minimum* MITRE *loss:* 15.38 *and* 14.91*; Minimum pairwise loss:* 34.10 *and* 33.93*.*

applications. In all our experiments, the initial weight $K_0$ is set such that the first iteration selects more than half the samples (as there are typically more easy samples than difficult ones). The weight is reduced by a factor $\mu = 1.3$ at each iteration and the parameters are initialized using $T_0 = 2$ iterations of the original CCCP algorithm.

## 5.1 Noun Phrase Coreference

**Problem Formulation.** Given the occurrence of all the nouns in a document, the goal of noun phrase coreference is to provide a clustering of the nouns such that each cluster refers to a single object. This task was formulated within the SSVM framework in [10] and extended to include latent variables in [23]. Formally, the input vector $\mathbf{x}$ consists of the pairwise features $\mathbf{x}_{ij}$ suggested in [16] between all pairs of noun phrases $i$ and $j$ in the document. The output $\mathbf{y}$ represents a clustering of the nouns. A hidden variable $\mathbf{h}$ specifies a forest over the nouns such that each tree in the forest consists of all the nouns of one cluster. Imputing the hidden variables involves finding the maximum spanning forest (which can be solved by Kruskal or Prims algorithm). Similar to [23], we employ two different loss functions, corresponding to the pairwise and MITRE scores.

**Dataset.** We use the publicly available MUC6 noun phrase coreference dataset, which consists of 60 documents. We use the same split of 30 training and 30 test documents as [23].

**Results.** We tested CCCP and our self-paced learning method on different values of $C$; the average training times over all 40 experiments (20 different values of $C$ and two different loss functions) for the two methods were 1183 and 1080 seconds respectively. Fig. 1 compares the two methods in terms of the value of the objective function (which is the main focus of this work), the loss over the training data and the loss over the test data. Note that self-paced learning significantly improves the objective function value in 11 of the 40 experiments (compared to only once when CCCP outperforms self-paced learning; see Fig. 1(a)). It also provides a better training and testing loss for both MITRE and pairwise scores when using the optimal value of $C$ (see Fig. 1(b)-(c)).

## 5.2 Motif Finding

**Problem Formulation.** We consider the problem of binary classification of DNA sequences, which was cast as a latent SSVM in [23]. Specifically, the input vector $\mathbf{x}$ consists of a DNA sequence of length $l$ (where each element of the sequence is a nucleotide of type A, G, T or C) and the output space $\mathcal{Y} = \{+1, -1\}$. In our experiments, the classes correspond to two different types of genes:

those that bind to a protein of interest with high affinity and those that do not. The positive sequences are assumed to contain particular patterns, called *motifs*, of length $m$ that are believed to be useful for classification. However, the starting position of the motif within a gene sequence is often not known. Hence, this position is treated as the hidden variable $\mathbf{h}$. For this problem, we use the joint feature vector suggested by [23]. Here, imputing the hidden variables simply involves a search for the starting position of the motif. The loss function $\Delta$ is the standard 0-1 classification loss.

**Dataset.**   We use the publicly available UniProbe dataset [4] that provides positive and negative DNA sequences for 177 proteins. For this work, we chose five proteins at random. The total number of sequences per protein is roughly $40,000$. For all the sequences, the motif length $m$ is known (provided with the UniProbe dataset) and the background Markov model is assumed to be of order $k = 3$. In order to specify a classification task for a particular protein, we randomly split the sequences into roughly 50% for training and 50% for testing.

(a) Objective function value

| | | | | | |
|---|---|---|---|---|---|
| CCCP | $92.77 \pm 0.99$ | $\mathbf{106.50 \pm 0.38}$ | $94.00 \pm 0.53$ | $116.63 \pm 18.78$ | $75.51 \pm 1.97$ |
| SPL | $\mathbf{92.37 \pm 0.65}$ | $106.60 \pm 0.30$ | $\mathbf{93.51 \pm 0.29}$ | $\mathbf{107.18 \pm 1.48}$ | $\mathbf{74.23 \pm 0.59}$ |

(b) Training error (%)

| | | | | | |
|---|---|---|---|---|---|
| CCCP | $27.10 \pm 0.44$ | $\mathbf{32.03 \pm 0.31}$ | $26.90 \pm 0.28$ | $34.89 \pm 8.53$ | $20.09 \pm 0.81$ |
| SPL | $\mathbf{26.94 \pm 0.26}$ | $32.04 \pm 0.23$ | $\mathbf{26.81 \pm 0.19}$ | $\mathbf{30.31 \pm 1.14}$ | $\mathbf{19.52 \pm 0.34}$ |

(c) Test error (%)

| | | | | | |
|---|---|---|---|---|---|
| CCCP | $27.10 \pm 0.36$ | $\mathbf{32.15 \pm 0.31}$ | $27.10 \pm 0.37$ | $35.42 \pm 8.19$ | $20.25 \pm 0.65$ |
| SPL | $\mathbf{27.08 \pm 0.38}$ | $32.24 \pm 0.25$ | $\mathbf{27.03 \pm 0.13}$ | $\mathbf{30.84 \pm 1.38}$ | $\mathbf{19.65 \pm 0.39}$ |

Table 1: *Mean and standard deviations for the motif finding experiments using the original* CCCP *algorithm (top row) and the proposed self-paced learning approach (bottom row). The better mean value is highlighted in bold. Note that self-paced learning provides an improved objective value (the primary concern of this work) for all proteins. The improvement in objective value also translates to an improvement in training and test errors.*

**Results.**   We used five different folds for each protein, randomly initializing the motif positions for all training samples using four different seed values (fixed for both methods). We report results for each method using the best seed (chosen according to the value of the objective function). For all experiments we use $C = 150$ and $\epsilon = 0.001$ (the large size of the dataset made cross-validation highly time consuming). The average time over all 100 runs for CCCP and self-paced learning are 824 and 1287 seconds respectively. Although our approach is slower than CCCP for this application, as table 1 shows, it learns a better set of parameters. While improvements for most folds are small, for the fourth protein, CCCP gets stuck in a bad local minimum despite using multiple random initializations (this is indicated by the large mean and standard deviation values). This behavior is to be expected: in many cases, the objective function landscape is such that CCCP avoids local optima; but in some cases, CCCP gets stuck in poor local optimum. Indeed, over all the 100 runs (5 proteins, 5 folds and 4 seed values) CCCP got stuck in a bad local minimum 18 times (where a bad local minimum is one that gave 50% test error) compared to 1 run where self-paced learning got stuck.

Fig. 2 shows the average Hamming distance between the motifs of the selected samples at each iteration of the self-paced learning algorithm. Note that initially the algorithm selects samples whose motifs have a low Hamming distance (which intuitively correspond to the easy samples for this application). It then gradually introduces more difficult samples (as indicated by the rise in the average Hamming distance). Finally, it considers all samples and attempts to find the most discriminative motif across the entire dataset. Note that the motifs found over the entire dataset using self-paced learning provide a smaller average Hamming distance than those found using the original CCCP algorithm, indicating a greater coherence for the resulting output.

### 5.3   Handwritten Digit Recognition

**Problem Formulation.**   Handwritten digit recognition is a special case of multi-label classification, and hence can be formulated within the SSVM framework. Specifically, given an input vector $\mathbf{x}$, which consists of $m$ grayscale values that represent an image of a handwritten digit, our aim is to predict the digit. In other words, $\mathcal{Y} = \{0, 1, \cdots, 9\}$. It is well-known that the accuracy of digit recognition can be greatly improved by explicitly modeling the deformations present in each image, for example see [18]. For simplicity, we assume that the deformations are restricted to an arbitrary rotation of the image, where the angle of rotation is not known beforehand. This angle (which takes a value from a finite discrete set) is modeled as the hidden variable $\mathbf{h}$. We specify the joint feature vector as $\Phi(\mathbf{x}, \mathbf{y}, \mathbf{h}) = (\mathbf{0}_{\mathbf{y}(m+1)}; \theta_{\mathbf{h}}(\mathbf{x}) \; 1; \mathbf{0}_{(9-\mathbf{y})(m+1)})$, where $\theta_{\mathbf{h}}(\mathbf{x})$ is the vector representation

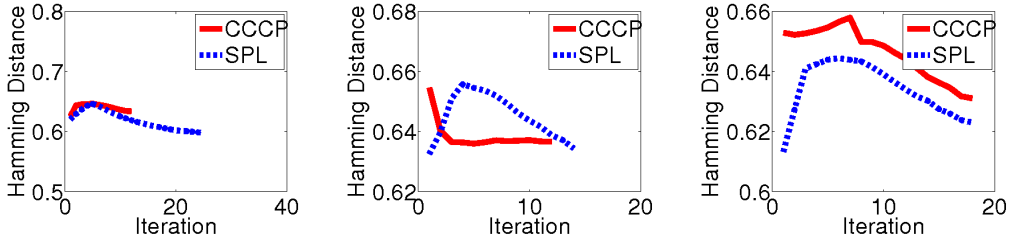

Figure 2: *Average Hamming distance between the motifs found in all selected samples at each iteration. Our approach starts with easy samples (small Hamming distance) and gradually introduces more difficult samples (large Hamming distance) until it starts to consider all samples of the training set. The figure shows results for three different protein-fold pairs. The average Hamming distance (over all proteins and folds) of the motifs obtained at convergence are* $0.6155$ *for* CCCP *and* $0.6099$ *for self-paced learning.*

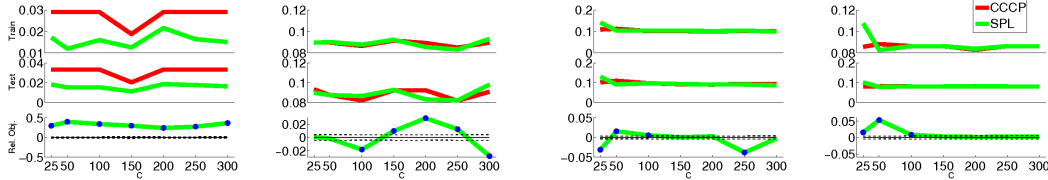

Figure 3: *Four digit pairs from* MNIST*: 1-7, 2-7, 3-8, 8-9. Relative objective is computed as in Fig. 1. Positive values indicate superior results for self-paced learning. The dotted black lines delineate where the difference is greater than the convergence criteria range ($C\epsilon$); differences outside this range are highlighted in blue.*

of the image $\mathbf{x}$ rotated by the angle corresponding to $\mathbf{h}$. In other words, the joint feature vector is the rotated image of the digit which is padded in the front and back with the appropriate number of zeroes. Imputing the hidden variables simply involves a search over a discrete set of angles. Similar to the motif finding experiment, we use the standard 0-1 classification loss.

**Dataset.** We use the standard MNIST dataset [15], which represents each handwritten digit as a vector of length 784 (that is, an image of size $28 \times 28$). For efficiency, we use PCA to reduce the dimensionality of each sample to 10. We perform binary classification on four difficult digit pairs (1-7, 2-7, 3-8, and 8-9), as in [25]. The training standard dataset size for each digit ranges from $5,851$ to $6,742$, and the test sets range from $974$ to $1,135$ digits. The rotation modeled by the hidden variable can take one of 11 discrete values, evenly spaced between $-60$ and $60$ degrees.

**Results.** For each digit pair, we use $C$ values ranging from 25 to 300, set $\epsilon = 0.001$, and set $K = \frac{10^4}{C}$. Modeling rotation as a hidden variable significantly improves classification performance, allowing the images to be better aligned with each other. Across all experiments for both learning methods, using hidden variables achieves better test error; the improvement over using no hidden variables is 12%, 8%, 11%, and 22%, respectively, for the four digit pairs. CCCP learning took an average of 18 minutes across all runs, while self-paced learning took an average of 53 minutes.

The above figure compares the training and test errors and objective values between CCCP and self-paced learning. Self-paced learning achieves significantly better values in 15 runs, and is worse in 4 runs, demonstrating that it helps find better solutions to the optimization problems. Though training and test errors do not necessarily correlate to objective values, the best test error across $C$ values is better for self-paced learning for one of the digit pairs (1-7), and is the same for the others.

## 5.4 Object Localization
**Problem Formulation.** Given a set of images along with labels that indicate the presence of a particular object category in the image (for example, a mammal), our goal is to learn discriminative object models for all object categories (that is, models that can distinguish between one object, say bison, from another, say elephant). In practice, although it is easy to mine such images from free photo-sharing websites such as Flickr, it is burdensome to obtain ground truth annotations of the exact location of the object in each image. To avoid requiring these human annotations, we model the location of objects as hidden variables. Formally, for a given image $\mathbf{x}$, category $\mathbf{y}$ and location $\mathbf{h}$, the score is modelled as $\mathbf{w}^T \Phi(\mathbf{x}, \mathbf{y}, \mathbf{h}) = \mathbf{w}_{\mathbf{y}}^T \Phi_{\mathbf{h}}(\mathbf{x})$, where $\mathbf{w}_{\mathbf{y}}$ are the parameters that corresponds to the class $\mathbf{y}$ and $\Phi_{\mathbf{h}}(\cdot)$ is the HOG [7, 9] feature extracted from the image at position $\mathbf{h}$ (the size of the object is assumed to be the same for all images — a reasonable assumption for our datasets). For

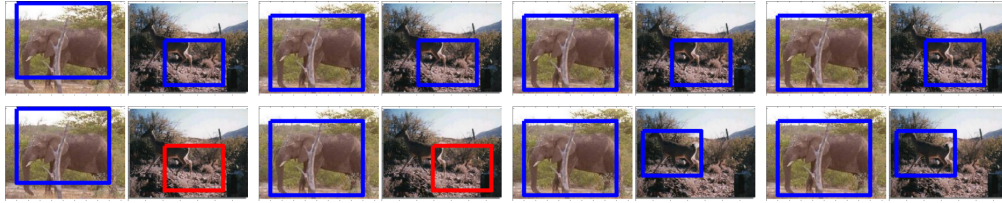

Figure 4: *The top row shows the imputed bounding boxes of an easy and a hard image using the* CCCP *algorithm over increasing iterations (left to right). Note that for the hard (deer) image, the bounding box obtained at convergence does not localize the object accurately. In contrast, the self-paced learning approach (bottom row) does not use the hard image during initial iterations (indicated by the red color of the bounding box). In subsequent iterations, it is able to impute accurate bounding boxes for both the easy and hard image.*

the above problem, imputing the hidden variables involves a simple search over possible locations in a given image. The loss function $\Delta(\mathbf{y}, \hat{\mathbf{y}})$ is again the standard 0-1 classification loss.

**Dataset.** We use images of 6 different mammals (approximately 45 images per mammal) that have been previously employed for object localization [13]. We split the images of each category into approximately 90% for training and 10% for testing.

**Results.** We use five different folds to compare our method with the state of the art CCCP algorithm. For each fold, we randomly initialized the location of the object in each image (the initialization was the same for the two methods). We used a value of $C = 10$ and $\epsilon = 0.001$. The average training time over all folds were 362 seconds and 482 seconds for CCCP and self-paced learning respectively. Table 2 shows the mean and standard deviation of three terms: the objective value, the training loss and the testing loss. Self-paced learning provided a significantly lower (more than tolerance) objective value than CCCP for all folds. The better objective value resulted in a substantial improvement in the training (for 4 folds) and testing loss (an improvement of approximately 4% for achieved for 2 folds). In these experiments, CCCP never outperformed self-paced learning for any of the three measures of performance.

| Objective | Train Loss (%) | Test Loss (%) | Objective | Train Loss (%) | Test Loss (%) |
|---|---|---|---|---|---|
| $4.70 \pm 0.11$ | $0.33 \pm 0.18$ | $16.92 \pm 5.16$ | $\mathbf{4.53 \pm 0.15}$ | $\mathbf{0.0 \pm 0.0}$ | $\mathbf{15.38 \pm 3.85}$ |

Table 2: *Results for the object localization experiment. Left:* CCCP. *Right: Self-paced learning. Note that self-paced learning provides better results for all measures of performance.*

Fig. 4 shows the imputed bounding boxes for two images during various iterations of the two algorithms. The proposed self-paced learning algorithm does not use the hard image during the initial iterations (as indicated by the red bounding box). In contrast, CCCP considers all images at each iteration. Note that self-paced learning provides a more accurate bounding box for the hard image at convergence, thereby illustrating the importance of learning in a meaningful order. In our experience, this was a typical behavior of the two algorithms.

## 6 Discussion

We proposed the self-paced learning regime in the context of parameter estimation for latent variable models. Our method works by iteratively solving a biconvex optimization problem that simultaneously selects easy samples and updates the parameters. Using four standard datasets from disparate domains (natural language processing, computational biology and computer vision) we showed that our method outperforms the state of the art approach.

In the current work, we solve the biconvex optimization problem using an alternate convex search strategy, which only provides us with a local minimum solution. Although our results indicate that such a strategy is more accurate than the state of the art, it is worth noting that the biconvex problem can also be solved using a global optimization procedure, for example the one described in [11]. This is a valuable direction for future work. We are also currently investigating the benefits of self-paced learning on other computer vision applications, where the ability to handle large and rapidly growing weakly supervised data is fundamental to the success of the field.

**Acknowledgements.** This work is supported by NSF under grant IIS 0917151, MURI contract N000140710747, and the Boeing company.

## References

[1] E. Allgower and K. Georg. *Numerical continuation methods: An introduction*. Springer-Verlag, 1990.

[2] M. Bazaraa, H. Sherali, and C. Shetty. *Nonlinear Programming - Theory and Algorithms*. John Wiley and Sons, Inc., 1993.

[3] Y. Bengio, J. Louradour, R. Collobert, and J. Weston. Curriculum learning. In *ICML*, 2009.

[4] M. Berger, G. Badis, A. Gehrke, and S. Talukder et al. Variation in homeodomain DNA binding revealed by high-resolution analysis of sequence preferences. *Cell*, 27, 2008.

[5] A. Blum and T. Mitchell. Combining labeled and unlabeled data with co-training. In *COLT*, 98.

[6] D. Cohn, Z. Ghahramani, and M. Jordan. Active learning with statistical models. *JAIR*, 4:129–145, 1996.

[7] N. Dalal and B. Triggs. Histograms of oriented gradients for human detection. In *CVPR*, 2005.

[8] A. Dempster, N. Laird, and D. Rubin. Maximum likelihood from incomplete data via the EM algorithm. *Journal of Royal Statistical Society*, 39(1):1–38, 1977.

[9] P. Felzenszwalb, D. McAllester, and D. Ramanan. A discriminatively trained, multiscale, deformable part model. In *CVPR*, 2008.

[10] T. Finley and T. Joachims. Supervised clustering with support vector machines. In *ICML*, 2005.

[11] C. Floudas and V. Visweswaran. Primal-relaxed dual global optimization approach. *Journal of Optimization Theory and Applications*, 78(2):187–225, 1993.

[12] A. Gelman, J. Carlin, H. Stern, and D. Rubin. *Bayesian Data Analysis*. Chapman and Hall, 1995.

[13] G. Heitz, G. Elidan, B. Packer, and D. Koller. Shape-based object localization for descriptive classification. *IJCV*, 2009.

[14] T. Joachims, T. Finley, and C.-N. Yu. Cutting-plane training for structural SVMs. *Machine Learning*, 77(1):27–59, 2009.

[15] Y. LeCun, L. Bottou, Y. Bengio, and P. Haffner. Gradient based learning applied to document recognition. *Proceedings of the IEEE*, 86(11):2278–2324, 1998.

[16] V. Ng and C. Cardie. Improving machine learning approaches to coreference resolution. In *ACL*, 2002.

[17] K. Nigam and R. Ghani. Analyzing the effectiveness and applicability of co-training. In *CIKM*, 2000.

[18] P. Simard, B. Victorri, Y. LeCun, and J. Denker. Tangent Prop - a formalism for specifying selected invariances in adaptive network. In *NIPS*, 1991.

[19] B. Sriperumbudur and G. Lanckriet. On the convergence of concave-convex procedure. In *NIPS Workshop on Optimization for Machine Learning*, 2009.

[20] B. Taskar, C. Guestrin, and D. Koller. Max-margin Markov networks. In *NIPS*, 2003.

[21] S. Tong and D. Koller. Support vector machine active learning with applications to text classification. *JMLR*, 2:45–66, 2001.

[22] I. Tsochantaridis, T. Hofmann, Y. Altun, and T. Joachims. Support vector machine learning for interdependent and structured output spaces. In *ICML*, 2004.

[23] C.-N. Yu and T. Joachims. Learning structural SVMs with latent variables. In *ICML*, 2009.

[24] A. Yuille and A. Rangarajan. The concave-convex procedure. *Neural Computation*, 15, 2003.

[25] K. Zhang, I. Tsang, and J. Kwok. Maximum margin clustering made practical. In *ICML*, 2007.

